# ALCOVE: A Connectionist Model of Human Category Learning

John K. Kruschke
Department of Psychology and Cognitive Science Program
Indiana University, Bloomington IN 47405-4201 USA
e-mail: kruschke@ucs.indiana.edu

## Abstract

ALCOVE is a connectionist model of human category learning that fits a broad spectrum of human learning data. Its architecture is based on well-established psychological theory, and is related to networks using radial basis functions. From the perspective of cognitive psychology, ALCOVE can be construed as a combination of exemplar-based representation and error-driven learning. From the perspective of connectionism, it can be seen as incorporating constraints into back-propagation networks appropriate for modelling human learning.

## 1 INTRODUCTION

ALCOVE is intended to accurately model human, perhaps non-optimal, performance in category learning. While it is a feed-forward network that learns by gradient descent on error, it is unlike standard back propagation (Rumelhart, Hinton & Williams, 1986) in its architecture, its behavior, and its goals. Unlike the standard back-propagation network, which was motivated by generalizing neuron-like perceptrons, the architecture of ALCOVE was motivated by a molar-level psychological theory, Nosofsky's (1986) generalized context model (GCM). The psychologically constrained architecture results in behavior that captures the detailed course of human category learning in many situations where standard back propagation fares less well. And, unlike most applications of standard back propagation, the goal of ALCOVE is not to discover new (hidden-layer) representations after lengthy training, but rather to model the course of learning itself (Kruschke, 1990c), by determining which dimensions of the given representation are most relevant to the task, and how strongly to associate exemplars with categories.

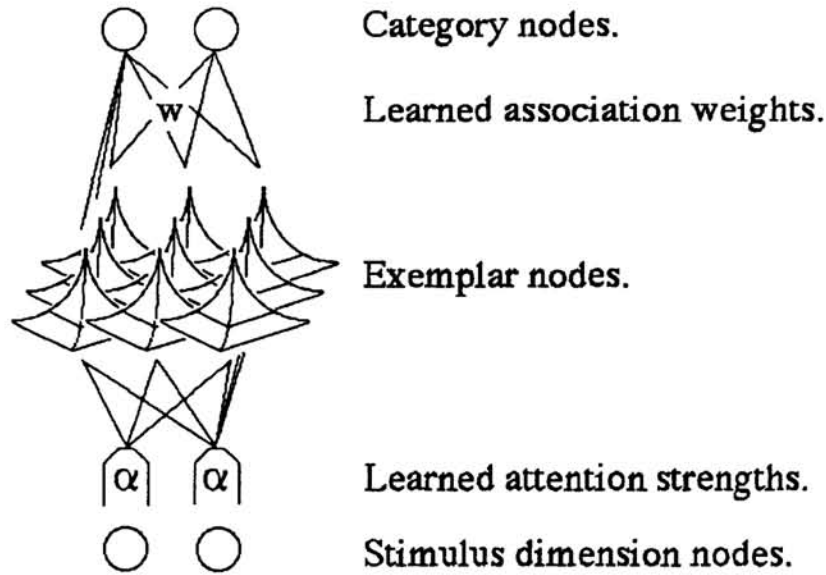

Category nodes.

Learned association weights.

Exemplar nodes.

Learned attention strengths.

Stimulus dimension nodes.

Figure 1: The architecture of ALCOVE (Attention Learning COVEring map). Exemplar nodes show their activation profile when $r = q = 1$ in Eqn. 1.

## 2  THE MODEL

Like the GCM, ALCOVE assumes that input patterns can be represented as points in a multi-dimensional psychological space, as determined by multi-dimensional scaling algorithms (e.g., Shepard, 1962). Each input node encodes a single psychological dimension, with the activation of the node indicating the value of the stimulus on that dimension. Figure 1 shows the architecture of ALCOVE, illustrating the case of just two input dimensions.

Each input node is gated by a dimensional *attention strength* $\alpha_i$. The attention strength on a dimension reflects the relevance of that dimension for the particular categorization task at hand, and the model learns to allocate more attention to relevant dimensions and less to irrelevant dimensions.

Each hidden node corresponds to a position in the multi-dimensional stimulus space, with one hidden node placed at the position of every training exemplar. Each hidden node is activated according to the psychological similarity of the stimulus to the exemplar represented by the hidden node. The similarity function comes from the GCM and the work of Shepard (1962; 1987): Let the position of the $j^{th}$ hidden node be denoted as $(h_{j1}, h_{j2}, \ldots)$, and let the activation of the $j^{th}$ hidden node be denoted as $a_j^{hid}$. Then

$$a_j^{hid} = \exp\left(-c\left(\sum_i \alpha_i |h_{ji} - a_i^{in}|^r\right)^{q/r}\right) \tag{1}$$

where $c$ is a positive constant called the *specificity* of the node, where the sum is taken over all input dimensions, and where $r$ and $q$ are constants determining the similarity metric and similarity gradient, respectively. For separable psychological

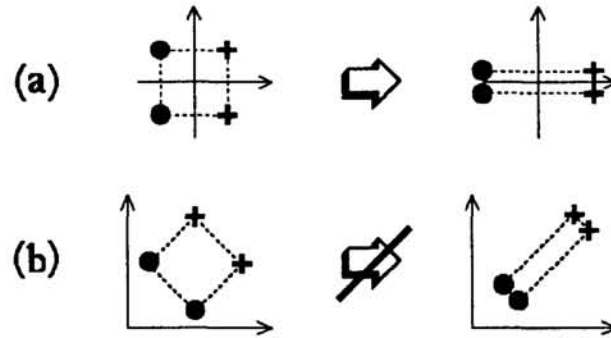

Figure 2: (a) Increasing attention on the horizontal axis and decreasing attention on the vertical axis causes exemplars of the two categories (denoted by dots and +'s) to have greater between-category dissimilarity and greater within-category similarity. (After Nosofsky, 1986, Fig. 2.)   (b) ALCOVE cannot differentially attend to diagonal axes.

dimensions, the city-block metric ($r = 1$) is used, while integral dimensions might call for a Euclidean metric ($r = 2$). An exponential similarity gradient ($q = 1$) is used here (Shepard, 1987; this volume), but a Gaussian similarity gradient ($q = 2$) can sometimes be appropriate.

The dimensional attention strengths adjust themselves so that exemplars from different categories become less similar, and exemplars within categories become more similar. Consider a simple case of four stimuli that form the corners of a square in input space, as in Figure 2(a). The two left stimuli are mapped to one category (indicated by dots) and the two right stimuli are mapped to another category (indicated by +'s). ALCOVE learns to increase the attention strength on the horizontal axis, and to decrease the attention strength on the vertical axis. On the other hand, ALCOVE cannot stretch or shrink diagonally, as suggested in Figure 2(b). This constraint is an accurate reflection of human performance, in that categories separated by a diagonal boundary tend to take longer to learn than categories separated by a boundary orthogonal to one dimension.

Each hidden node is connected to output nodes that correspond to response categories. The connection from the $j^{th}$ hidden node to the $k^{th}$ category node has a connection weight denoted $w_{kj}$, called the *association weight* between the exemplar and the category. The output (category) nodes are activated by the linear rule used in the GCM and the network models of Gluck and Bower (1988a,b):

$$a_k^{out} = \sum_{\substack{hid \\ j}} w_{kj} a_j^{hid} . \tag{2}$$

In ALCOVE, unlike the GCM, the association weights are learned and can take on any real value, including negative values. Category activations are mapped to response probabilities using the same choice rule as was used in the GCM and network models. Thus,

$$\Pr(K) = \exp(\phi\, a_K^{out}) \Big/ \sum_{\substack{out \\ k}} \exp(\phi\, a_k^{out}) \tag{3}$$

where $\phi$ is a real-valued scaling constant. In other words, the probability of classifying the given stimulus into category $K$ is determined by the magnitude of category $K$'s activation relative to the sum of all category activations.

The dimensional attention strengths, $\alpha_i$, and the association weights, $w_{kj}$, are learned by gradient descent on sum-squared error, as used in standard back propagation (Rumelhart et al., 1986) and in the network models of Gluck and Bower (1988a,b). Details can be found in Kruschke (1990a,b). In fitting ALCOVE to human learning data, there are four free parameters: the fixed specificity $c$ in Equation 1; the probability mapping constant $\phi$ in Equation 3; the association weight learning rate; and, the attention strength learning rate.

In summary, ALCOVE extends Nosofsky's (1986) GCM by having a learning mechanism and by allowing any positive or negative values for association weights, and it extends Gluck and Bower's (1988a,b) network models by including explicit attention strengths and by using continuous input dimensions. It is a combination of exemplar-based category representations with error-driven learning, as alluded to by Estes et al. (1989; see also Hurwitz, 1990). ALCOVE can also be construed as a form of (non-)radial basis function network, if $r = q = 2$ in Equation 1. In the form described here, the hidden nodes are placed at positions where training exemplars occur, but another option, described by Kruschke (1990a,b), is to scatter hidden nodes over the input space to form a covering map. Both these methods work well in fitting human data in some situations, but the exemplar-based approach has advantages (Kruschke, 1990a,b). ALCOVE can also be compared to a standard back-propagation network that has adaptive attentional multipliers on its input nodes (cf. Mozer and Smolensky, 1989), but with fixed input-to-hidden weights (Kruschke 1990b, p.33). Such a network behaves similarly to a covering-map version of ALCOVE. Moreover, such back-prop networks are susceptible to catastrophic retroactive interference (Ratcliff, 1990; McCloskey & Cohen, 1989), unlike ALCOVE.

## 3    APPLICATIONS

Several applications of ALCOVE to modelling human performance are detailed elsewhere (Kruschke, 1990a,b); a few will be summarized here.

### 3.1    RELATIVE DIFFICULTY OF CATEGORY STRUCTURES

The classic work of Shepard, Hovland and Jenkins (1961) explored the relative difficulty of learning different category structures. As a simplified example, the linearly separable categories in Figure 2(a) are easier to learn than the exclusive-or problem (which would have the top-left and bottom-right exemplars mapped to one category, and the top-right and bottom-left mapped to the other). Shepard et al. carefully considered several candidate explanations for the varying difficulties, and concluded that some form of attentional learning was necessary to account for their results. That is, people seemed to be able to determine which dimensions were relevant or irrelevant, and they allocated attention to dimensions accordingly. Category structures with fewer relevant dimensions were easier to learn. ALCOVE has just the sort of attentional learning mechanism called for, and can match the relative difficulties observed by Shepard et al.

## 3.2   BASE-RATE NEGLECT

A recent series of experiments (Gluck & Bower, 1988b; Estes *et al.*, 1989; Shanks, 1990; Nosofsky *et al.*, 1991) investigated category learning when the assignment of exemplars to categories was probabilistic and the base rates of the categories were unequal. In these experiments, there were two categories (one "rare" and the other "common") and four binary-valued stimulus dimensions. The stimulus values were denoted s1 and s1* for the first dimension, s2 and s2* for the second dimension, and so on. The probalities were arranged such that over the course of training, the normative probability of each category, given s1 alone, was 50%. However, when presented with feature s1 alone, human subjects classified it as the rare category significantly more than 50% of the time. It was as if people were neglecting the base rates of the categories.

Gluck and Bower (1988b) and Estes *et al.* (1989) compared two candidate models to account for the apparent base-rate neglect. One was a simple exemplar-based model that kept track of each training exemplar, and made predictions of categorizations by summing up frequencies of occurence of each stimulus value for each category. The exemplar-based model was unable to predict base-rate neglect. The second model they considered, the "double-node network," was a one-layer error-driven network that encoded each binary-valued dimension with a pair of input nodes. The double-node model was able show base-rate neglect.

ALCOVE is an exemplar-based model, and so it is challenged by those results. In fact, Kruschke (1990a,b) and Nosofsky *et al.* (1991) show that ALCOVE fits the trail-by-trial learning and base-rate neglect data as well as or better than the double-node model.

## 3.3   THREE-STAGE LEARNING OF RULES AND EXCEPTIONS

One of the best-known connectionist models of human learning is Rumelhart and McClelland's (1986) model of verb past tense acquistion. One of the main phenomena they wished to model was three-stage learning of irregular verbs: First a few high-frequency irregulars are learned; second, many regular verbs are learned with some interference to the previously learned irregulars; and third, the high-frequency irregulars are re-learned.[1] In order to reproduce three-stage learning in their model, Rumelhart and McClelland had to change the training corpus during learning, so that early on the network was trained with ten verbs, 80% of which were irregular, and later the network was trained with 420 verbs, only 20% of which were irregular. It remains a challenge to connectionist models to show three-stage learning of rules and exceptions while keeping the training set constant.

While ALCOVE has not been applied to the verb-learning situation (and perhaps should not be, as a multi-dimensional similarity-space might not be a tractable representation for verbs), it can show three-stage learning of rules and exceptions in simpler but analogous situations. Figure 3 shows an arrangement of training exemplars, most of which can be classified by the simple rule, "if it's to the right

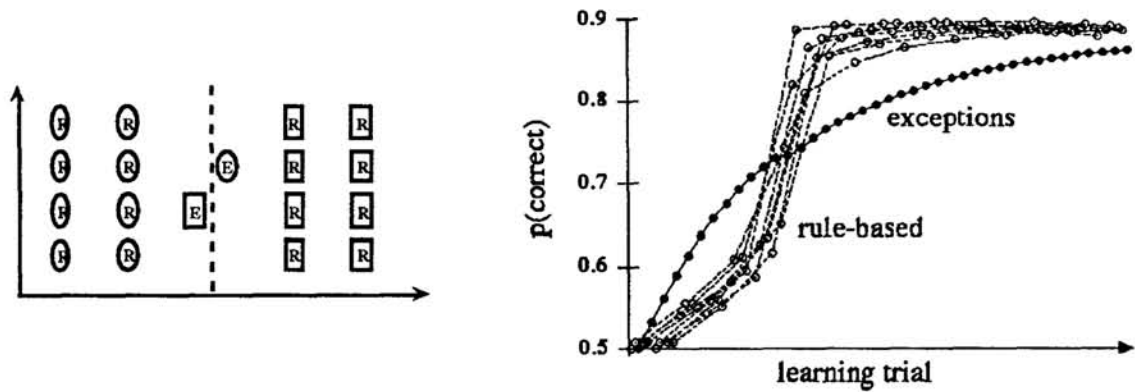

Figure 3: Left panel shows arrangement of rule-following (R) and exceptional (E) cases. Right panel shows the performance of ALCOVE. The ratio of E to R cases and all parameters of the model were fixed throughout training.

of the dashed line, then it's in the 'rectangle' category, otherwise it's in the 'oval' category." The rule-following cases are marked with an "R." There are two exceptional cases near the dashed line, marked with an "E." Exceptional exemplars occurred 4 times as often as rule-following exemplars. The right panel of Figure 3 shows that ALCOVE initially learns the E cases better than the R cases, but that later in learning the R cases surpass the E's. The reason is that early in learning, ALCOVE is primarily building up association weights and has not yet shifted much attention away from the irrelevant dimension. Associations from the E cases grow more quickly because they are more frequent. Once the associations are established, then there is a basis for attention to be shifted away from the irrelevant dimension, rapidly improving performance on the R cases. At the time of this writing, these results have the status of a provocative demonstration, but experiments with human subjects in similar learning situations are presently being undertaken.

## Acknowledgment

This research was supported in part by Biomedical Research Support Grant RR 7031-25 from the National Institutes of Health.

## Footnotes

[1] There is evidence that three-stage learning is only very subtle in verb past tense acquisition (e.g., Marcus, 1990), but whether it exists more robustly in the simpler category learning domains addressed by ALCOVE is still an open question.

## References

Estes, W. K., Campbell, J. A., Hatsopoulos, N., & Hurwitz, J. B. (1989). Base-rate effects in category learning: A comparison of parallel network and memory storage-retrieval models. *J. Exp. Psych. Learning, Memory and Cognition*, **15**, 556-576.

Gluck, M. A. & Bower, G. H. (1988a). Evaluating an adaptive network model of human learning. *J. of Memory and Language*, **27**, 166-195.

Gluck, M. A. & Bower, G. H. (1988b). From conditioning to category learning: An adaptive network model. *J. Exp. Psych. General*, **117**, 227-247.

Hurwitz, J. B. (1990). A hidden-pattern unit network model of category learning. Doctoral dissertation, Harvard University.

Kruschke, J. K. (1990a). A connectionist model of category learning. Doctoral dissertation, University of California at Berkeley. Available from University Microfilms International.

Kruschke, J. K. (1990b). ALCOVE: A connectionist model of category learning. Research Report 19, Cognitive Science Program, Indiana University.

Kruschke, J. K. (1990c). How connectionist models learn: The course of learning in connectionist networks. *Behavioral and Brain Sciences*, **13**, 498-499.

Marcus, G. F., Ullman, M., Pinker, S., Hollander, M., Rosen, T. J., & Xu, F. (1990). Overregularization. Occasional Paper #41, MIT Center for Cognitive Science.

McCloskey, M. & Cohen, N. J. (1989). Catastrophic interference in connectionist networks: the sequential learning problem. In: G. Bower (ed.), *The Psychology of Learning and Motivation, Vol. 24*. New York: Academic Press.

Mozer, M. C., & Smolensky, P. (1989). Skeletonization: A technique for trimming the fat from a network via relevance assessment. In: D. S. Touretzky (ed.), *Advances in Neural Information Processing Systems, I*, pp. 107-115. San Mateo, CA: Morgan Kaufmann.

Nosofsky, R. M. (1986). Attention, similarity and the identification-categorization relationship. *J. Exp. Psych. General*, **115**, 39-57.

Nosofsky, R. M., Kruschke, J. K., & McKinley, S. (1991). Comparisons between adaptive network and exemplar models of classification learning. Research Report 35, Cognitive Science Program, Indiana University.

Ratcliff, R. (1990). Connectionist models of recognition memory: Constraints imposed by learning and forgetting functions. *Psychological Review*, **97**, 285-308.

Rumelhart, D. E., Hinton, G. E., & Williams, R. J. (1986). Learning internal representations by back-propagating errors. In: D. E. Rumelhart & J. L. McClelland (eds.), *Parallel Distributed Processing, Vol. 1*, pp. 318-362. Cambridge, MA: MIT Press.

Rumelhart, D. E., & McClelland, J. L. (1986). On learning the past tenses of english verbs. In: J. L. McClelland & D. E. Rumelhart (eds.), *Parallel Distributed Processing, Vol. 2*, pp. 216-271. Cambridge, MA: MIT Press.

Shanks, D. R. (1990). Connectionism and the learning of probabilistic concepts. *Quarterly J. Exp. Psych.*, **42A**, 209-237.

Shepard, R. N. (1962). The analysis of proximities: Multidimensional scaling with an unknown distance function, I & II. *Psychometrika*, **27**, 125-140, 219-246.

Shepard, R. N. (1987). Toward a universal law of generalization for psychological science. *Science*, **237**, 1317-1323.

Shepard, R. N., Hovland, C. L., & Jenkins, H. M. (1961). Learning and memorization of classifications. *Psychological Monographs*, **75**(13), Whole No. 517.
